# THE SIGMOID NONLINEARITY IN PREPYRIFORM CORTEX

Frank H. Eeckman
University of California, Berkeley, CA 94720

## ABSTRACT

We report a study on the relationship between EEG amplitude values and unit spike output in the prepyriform cortex of awake and motivated rats. This relationship takes the form of a sigmoid curve, that describes normalized pulse-output for normalized wave input. The curve is fitted using nonlinear regression and is described by its slope and maximum value.

Measurements were made for both excitatory and inhibitory neurons in the cortex. These neurons are known to form a monosynaptic negative feedback loop. Both classes of cells can be described by the same parameters.

The sigmoid curve is asymmetric in that the region of maximal slope is displaced toward the excitatory side. The data are compatible with Freeman's model of prepyriform burst generation. Other analogies with existing neural nets are being discussed, and the implications for signal processing are reviewed. In particular the relationship of sigmoid slope to efficiency of neural computation is examined.

## INTRODUCTION

The olfactory cortex of mammals generates repeated nearly sinusoidal bursts of electrical activity (EEG) in the 30 to 60 Hz. range[1]. These bursts ride on top of a slower ( 1 to 2 Hz.), high amplitude wave related to respiration. Each burst begins shortly after inspiration and terminates during expiration. They are generated locally in the cortex. Similar bursts occur in the olfactory bulb (OB) and there is a high degree of correlation between the activity in the two structures[1].

The two main cell types in the olfactory cortex are the superficial pyramidal cell (type A), an excitatory neuron receiving direct input from the OB, and the cortical granule cell (type B), an inhibitory interneuron. These cell groups are monosynaptically connected in a negative feedback loop[2].

Superficial pyramidal cells are mutually excitatory[3, 4, 5] as well as being excitatory to the granule cells. The granule cells are inhibitory to the pyramidal cells as well as to each other[3, 4, 6].

In this paper we focus on the analysis of amplitude dependent properties: How is the output of a cellmass (pulses) related to the synaptic potentials (ie. waves)? The concurrent recording of multi-unit spikes and EEG allows us to study these phenomena in the olfactory cortex.

The anatomy of the olfactory system has been extensively studied beginning with the work of S. Ramon y Cajal [7]. The regular geometry and the simple three-layered architecture makes these structures ideally suitable for EEG recording [4, 8]. The EEG generators in the various olfactory regions have been identified and their synaptic connectivities have been extensively studied[9, 10, 5, 4, 11, 6].

The EEG is the scalar sum of synaptic currents in the underlying cortex. It can be recorded using low impedance (< .5 Mohm) cortical or depth electrodes. Multiunit signals are recorded in the appropriate cell layers using high impedance (> .5 Mohm) electrodes and appropriate high pass filtering.

Here we derive a function that relates waves (EEG) to pulses in the olfactory cortex of the rat. This function has a sigmoidal shape. The derivative of this curve

gives us the gain curve for wave-to-pulse conversion. This is the forward gain for neurons embedded in the cortical cellmass. The product of the forward gain values of both sets of neurons (excitatory and inhibitory) gives us the feedback gain values. These ultimately determine the dynamics of the system under study.

## MATERIALS AND METHODS

A total of twenty-nine rats were entered in this study. In each rat a linear array of 6 100 micron stainless steel electrodes was chronically implanted in the prepyriform (olfactory) cortex. The tips of the electrodes were electrolytically sharpened to produce a tip impedance on the order of .5 to 1 megaohm. The electrodes were implanted laterally in the midcortex, using stereotaxic coordinates. Their position was verified electrophysiologically using a stimulating electrode in the olfactory tract. This procedure has been described earlier by Freeman 12. At the end of the recording session a small iron deposit was made to help in histological verification. Every electrode position was verified in this manner.

Each rat was recorded from over a two week period following implantation. All animals were awake and attentive. No stimulation (electrical or olfactory) was used. The background environment for recording was the animal's home cage placed in the same room during all sessions.

For the present study two channels of data were recorded concurrently. Channel 1 carried the EEG signal, filtered between 10 and 300 Hz. and digitized at 1 ms intervals. Channel 2 carried standard pulses 5 V, 1.2 ms wide, that were obtained by passing the multi-unit signal (filtered between 300 Hz. and 3kHz.) through a window discriminator.

These two time-series were stored on disk for off-line processing using a Perkin-Elmer 3220 computer. All routines were written in FORTRAN. They were tested on data files containing standard sine-wave and pulse signals.

## DATA PROCESSING

The procedures for obtaining a two-dimensional conditional pulse probability table have been described earlier 4. This table gives us the probability of occurrence of a spike conditional on both time and normalized EEG amplitude value.

By counting the number of pulses at a fixed time-delay, where the EEG is maximal in amplitude, and plotting them versus the normalized EEG amplitudes, one obtains a sigmoidal function: The Pulse probability Sigmoid Curve (PSC) 13, 14. This function is normalized by dividing it by the average pulse level in the record. It is smoothed by passing it through a digital 1:1:1 filter and fitted by nonlinear regression.

The equations are:

$$Q = Q_{max} ( 1 - \exp [ - ( e^v - 1 ) / Q_{max} ] ) \quad \text{for } v > -u_0 \quad (1)$$
$$Q = -1 \quad \text{for } v < -u_0$$

where $u_0$ is the steady state voltage, and $Q = (p-p_0)/p_0$.
  and $Q_{max} = (p_{max}-p_0)/p_0$.
  $p_0$ is the background pulse count, $p_{max}$ is the maximal pulse count.
These equations rely on one parameter only. The derivation and justification for these equations were discussed in an earlier paper by Freeman 13.

## RESULTS

Data were obtained from all animals. They express normalized pulse counts, a dimensionless value as a function of normalized EEG values, expressed as a Z-score (ie. ranging from - 3 sd. to + 3 sd., with mean of 0.0). The true mean for the EEG after filtering is very close to 0.0 mV and the distribution of amplitude values is very nearly Gaussian.

The recording convention was such that high EEG-values ( ie. > 0.0 to + 3.0 sd.) corresponded to surface-negative waves. These in turn occur with activity at the apical dendrites of the cells of interest. Low EEG values (ie. from - 3.0 sd. to < 0.0) corresponded to surface-positive voltage values, representing inhibition of the cells.

The data were smoothed and fitted with equation (1). This yielded a $Q_{max}$ value for every data file. There were on average 5 data files per animal. Of these 5, an average of 3.7 per animal could be fitted succesfully with our technique. In 25 % of the traces, each representing a different electrode pair, no correlations between spikes and the EEG were found.

Besides $Q_{max}$ we also calculated Q' the maximum derivative of the PSC, representing the maximal gain.

There were 108 traces in all. In the first 61 cases the $Q_{max}$ value described the wave-to-pulse conversion for a class of cells whose maximum firing probability is in phase with the EEG. These cells were labelled type A cells [2]. These traces correspond to the excitatory pyramidal cells. The mean for $Q_{max}$ in that group was 14.6, with a standard deviation of 1.84. The range was 10.5 to 17.8.

In the remaining 47 traces the $Q_{max}$ described the wave-to-pulse conversion for class B cells. Class B is a label for those cells whose maximal firing probability lags the EEG maximum by approximately 1/4 cycle. The mean for $Q_{max}$ in that group was 14.3, with a standard deviation of 2.05. The range in this group was 11.0 to 18.8.

The overall mean for $Q_{max}$ was 14.4 with a standard deviation of 1.94. There is no difference in $Q_{max}$ between both groups as measured by the Student t-test. The nonparametric Wilcoxon rank-sum test also found no difference between the groups ( p = 0.558 for the t-test; p = 0.729 for the Wilcoxon).

Assuming that the two groups have $Q_{max}$ values that are normally distributed ( in group A, mean = 14.6, median = 14.6; in group B, mean = 14.3, median = 14.1), and that they have equal variances ( st. deviation group A is 1.84; st. deviation group B is 2.05) but different means, we estimated the power of the t-test to detect that difference in means.

A difference of 3 points between the $Q_{max}$'s of the respective groups was considered to be physiologically significant. Given these assumptions the power of the t-test to detect a 3 point difference was greater than .999 at the alpha .05 level for a two sided test. We thus feel reasonably confident that there is no difference between the $Q_{max}$ values of both groups.

The first derivative of the PSC gives us the gain for wave-to-pulse conversion[4]. The maximum value for this first derivative was labelled Q'. The location at which the maximum Q' occurs was labelled $V_{max}$. $V_{max}$ is expressed in units of standard deviation of EEG amplitudes.

The mean for Q' in group A was 5.7, with a standard deviation of .67, in group B it was 5.6 with standard deviation of .73. Since Q' depends on $Q_{max}$, the same statistics apply to both: there was no significant difference between the two groups for slope maxima.

## Figure 1. Distribution of Qmax values

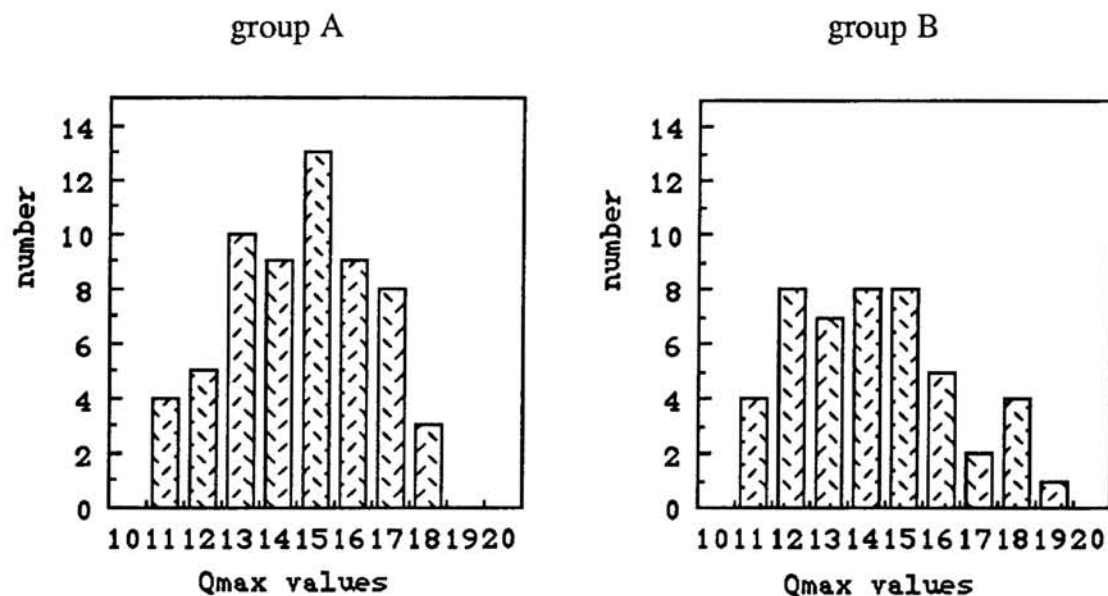

group A                    group B

The mean for $V_{max}$ was at 2.15 sd. +/- .307. In every case $V_{max}$ was on the excitatory side from 0.00, ie. at a positive value of EEG Z-scores. All values were greater than 1.00. A similar phenomenon has been reported in the olfactory bulb 4, 14, 15.

## Figure 2. Examples of sigmoid fits.

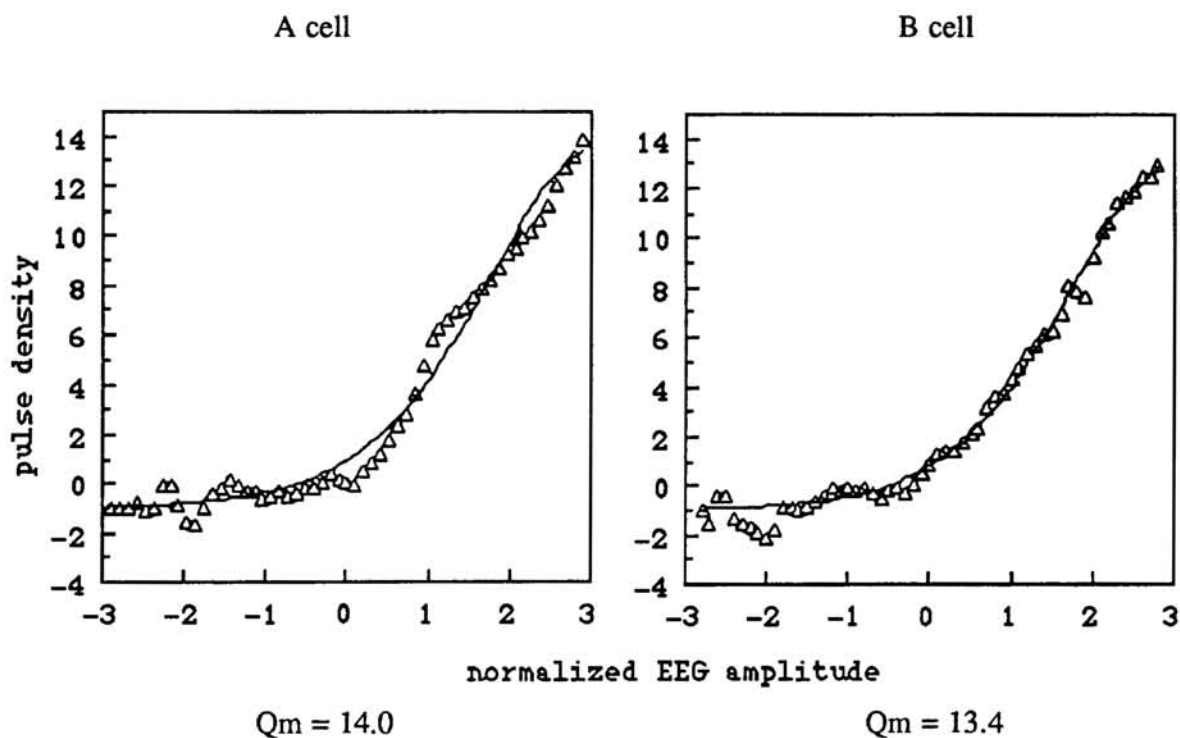

A cell                    B cell

Qm = 14.0                Qm = 13.4

## COMPARISON WITH DATA FROM THE OB

Previously we derived $Q_{max}$ values for the mitral cell population in the olfactory bulb[14]. The mitral cells are the output neurons of the bulb and their axons form the lateral olfactory tract (LOT). The LOT is the main input to the pyramidal cells (type A) in the cortex.

For awake and motivated rats (N = 10) the mean $Q_{max}$ value was 6.34 and the standard deviation was 1.46. The range was 4.41- 9.53. For anesthetized animals (N= 8) the mean was 2.36 and the standard deviation was 0.89. The range was 1.15-3.62. There was a significant difference between anesthetized and awake animals. Furthermore there is a significant difference between the $Q_{max}$ value for cortical cells and the $Q_{max}$ value for bulbar cells ( non - overlapping distributions).

## DISCUSSION

An important characteristic of a feedback loop is its feedback gain. There is ample evidence for the existence of feedback at all levels in the nervous system. Moreover specific feedback loops between populations of neurons have been described and analyzed in the olfactory bulb and the prepyriform cortex [3, 9, 4].

A monosynaptic negative feedback loop has been shown to exist in the PPC, between the pyramidal cells and inhibitory cells, called granule cells [3, 2, 6, 16]. Time series analysis of concurrent pulse and EEG recordings agrees with this idea.

The pyramidal cells are in the forward limb of the loop: they excite the granule cells. They are also mutually excitatory [2,4,16]. The granule cells are in the feedback limb: they inhibit the pyramidal cells. Evidence for mutual inhibition (granule to granule) in the PPC also exists [17, 6].

The analysis of cell firings versus EEG amplitude at selected time-lags allows one to derive a function (the PSC) that relates synaptic potentials to output in a neural feedback system. The first derivative of this curve gives an estimate of the forward gain at that stage of the loop. The procedure has been applied to various structures in the olfactory system [4, 13, 15, 14]. The olfactory system lends itself well to this type of analysis due to its geometry, topology and well known anatomy.

Examination of the experimental gain curves shows that the maximal gain is displaced to the excitatory side. This means that not only will the cells become activated by excitatory input, but their mutual interaction strength will increase. The result is an oscillatory burst of high frequency ( 30- 60 Hz.) activity. This is the mechanism behind bursting in the olfactory EEG [4, 13].

In comparison with the data from the olfactory bulb one notices that there is a significant difference in the slope and the maximum of the PSC. In cortex the values are substantially higher, however the $V_{max}$ is similar. C. Gray [15] found a mean value of 2.14 +/- 0.41 for $V_{max}$ in the olfactory bulb of the rabbit (N= 6). Our value in the present study is 2.15 +/- .31. The difference is not statistically significant.

There are important aspects of nonlinear coupling of the sigmoid type that are of interest in cortical functioning. A sigmoid interaction between groups of elements ("neurons") is a prominent feature in many artificial neural nets. S. Grossberg has extensively studied the many desirable properties of sigmoids in these networks. Sigmoids can be used to contrast-enhance certain features in the stimulus. Together with a thresholding operation a sigmoid rule can effectively quench noise. Sigmoids can also provide for a built in gain control mechanism [18, 19].

Changing sigmoid slopes have been investigated by J. Hopfield. In his network changing the slope of the sigmoid interaction between the elements affects the number of attractors that the system can go to 20. We have previously remarked upon the similarities between this and the change in sigmoid slope between waking and anesthetized animals 14. Here we present a system with a steep slope (the PPC) in series with a system with a shallow slope (the OB).

Present investigations into similarities between the olfactory bulb and Hopfield networks have been reported 21, 22. Similarities between the cortex and Hopfield-like networks have also been proposed 23.

Spatial amplitude patterns of EEG that correlate with significant odors exist in the bulb 24. A transmission of "wave-packets" from the bulb to the cortex is known to occur 25. It has been shown through cofrequency and phase analysis that the bulb can drive the cortex 25, 26. It thus seeems likely that spatial patterns may also exist in the cortex. A steeper sigmoid, if the analogy with neural networks is correct, would allow the cortex to further classify input patterns coming from the olfactory bulb.

In this view the bulb could form an initial classifier as well as a scratch-pad memory for olfactory events. The cortex could then be the second classifier, as well as the more permanent memory.

These are at present speculations that may turn out to be premature. They nevertheless are important in guiding experiments as well as in modelling. Theoretical studies will have to inform us of the likelihood of this kind of processing.

## REFERENCES

1 S.L. Bressler and W.J. Freeman, Electroencephalogr. Clin. Neurophysiol. **50** : 19 (1980).
2 W.J. Freeman, J. Neurophysiol. **31**: 1 (1968).
3 W.J. Freeman, Exptl. Neurol. **10**: 525 (1964).
4 W.J. Freeman, Mass Action in the Nervous System. (Academic Press, N.Y., 1975), Chapter 3.
5 L.B. Haberly and G.M. Shepherd, Neurophys. **36**: 789 (1973).
6 L.B. Haberly and J.M. Bower, J. Neurophysiol. **51**: 90 (1984).
7 S. Ramon y Cajal, Histologie du Systeme Nerveux de l'Homme et des Vertebres. ( Ed. Maloine, Paris, 1911) .
8 W.J. Freeman, Biol. Cybernetics. **35**: 21 (1979).
9 W. Rall and G.M. Shepherd, J. Neurophysiol. **31**: 884 (1968).
10 G.M. Shepherd, Physiol. Rev. **52**: 864 (1972).
11 L.B. Haberly and J.L. Price, J. Comp. Neurol. **178**; 711 (1978).
12 W.J. Freeman, Exptl. Neurol. **6**: 70 (1962).
13 W.J. Freeman, Biol. Cybernetics. **33**: 237 (1979).
14 F.H. Eeckman and W.J. Freeman, AIP Proc. **151**: 135 (1986).
15 C.M. Gray, Ph.D. thesis, Baylor College of Medicine (Houston,1986)
16 L.B. Haberly, Chemical Senses, **10**: 219 (1985).
17 M. Satou et al., J. Neurophysiol. **48**: 1157 (1982).
18 S. Grossberg, Studies in Applied Mathematics, Vol LII, 3 (MIT Press, 1973) p 213.
19 S. Grossberg, SIAM-AMS Proc. **13**: 107 (1981).
20 J.J Hopfield, Proc. Natl. Acad. Sci. USA **81**: 3088 (1984).
21 W.A. Baird, Physica **22D**: 150 (1986).
22 W.A. Baird, AIP Proceedings **151**: 29 (1986).
23 M. Wilson and J. Bower, Neurosci. Abstr. **387.10** (1987).

24 K.A. Grajski and W.J. Freeman, AIP Proc. **151**: 188 (1986).
25 S.L. Bressler, Brain Res. **409**: 285 (1986).
26 S.L. Bressler, Brain Res. **409**: 294 (1986).
